# Hierarchical Learning of Dimensional Biases in Human Categorization

**Katherine Heller**
Department of Engineering
University of Cambridge
Cambridge CB2 1PZ
heller@gatsby.ucl.ac.uk

**Adam Sanborn**
Gatsby Computational Neuroscience Unit
University College London
London WC1N 3AR
asanborn@gatsby.ucl.ac.uk

**Nick Chater**
Cognitive, Perceptual and Brain Sciences
University College London
London WC1E 0AP
n.chater@ucl.ac.uk

## Abstract

Existing models of categorization typically represent to-be-classified items as points in a multidimensional space. While from a mathematical point of view, an infinite number of basis sets can be used to represent points in this space, the choice of basis set is psychologically crucial. People generally choose the same basis dimensions – and have a strong preference to generalize along the axes of these dimensions, but not "diagonally". What makes some choices of dimension special? We explore the idea that the dimensions used by people echo the natural variation in the environment. Specifically, we present a rational model that does not assume dimensions, but learns the same type of dimensional generalizations that people display. This bias is shaped by exposing the model to many categories with a structure hypothesized to be like those which children encounter. The learning behaviour of the model captures the developmental shift from roughly "isotropic" for children to the axis-aligned generalization that adults show.

## 1 Introduction

Given only a few examples of a particular category, people have strong expectations as to which new examples also belong to that same category. These expectations provide important insights into how objects are mentally represented. One basic insight into mental representations is that objects that have similar observed properties will be expected to belong to the same category, and that expectation decreases as the Euclidean distance between the properties of the objects increases [1, 2].

The Euclidean distance between observed properties is only part of the story however. Dimensions also play a strong role in our expectations of categories. People do not always generalize isotropically: the *direction* of generalizations turns out to be centrally important. Specifically, people generalize along the dimensions, such as size, color, or shape – dimensions that are termed *separable*. In contrast, dimensions such as hue and saturation, which show isotropic generalization are termed *integral* [3] . An illustration of the importance of separable dimensions is found in the time to learn categories. If dimensions did not play a strong role in generalization, then rotating a category structure in a parameter space of separable dimensions should not influence how easily it can be learned. To the contrary, rotating a pair of categories 45 degrees [3, 4] makes it more difficult to learn to

discriminate between them. Similarity rating results also show strong trends of judging objects to be more similar if they match along separable dimensions [3, 5].

The tendency to generalize categories along separable dimensions is learned over development. On dimensions such as size and color, children produce generalizations that are more isotropic than adults [6]. Interestingly, the developmental transition between isotropic and dimensionally biased generalizations is gradual [7].

What privileges separable dimensions? And why are they acquired over development? One possibility is that there is corresponding variation in real-world categories, and this provides a bias that learners carry over to laboratory experiments. For example, Rosch et al. [1] identified shape as a key constant in categories, and we can find categories that are constant along other separable dimensions as well. For instance, categories of materials such as gold, wood, and ice all display a characteristic color while being relatively unconstrained as to the shapes and sizes that they take. Size is often constrained in artifacts such as books and cars, while color can vary across a very wide range.

Models of categorization are able to account for both the isotropic and dimension-based components of generalization. Classic models of categorization, such as the exemplar and prototype model, account for these using different mechanisms [8, 9, 10]. Rational models of categorization have accounted for dimensional biases by assuming that the shapes of categories are aligned with the axes that people use for generalization [11, 12, 13]. Neither the classic models nor rational models have investigated how people learn to use the particular dimension basis that they do.

This paper presents a model that learns the dimensional basis that people use for generalization. We connect these biases with a hypothesis about the structure of categories in the environment and demonstrate how exposure to these categories during development results in human dimensional biases. In the next section, we review models of categorization and how they have accounted for dimensional biases. Next, we review current nonparametric Bayesian models of categorization, which all require that the dimensions be hand-coded. Next, we introduce a new prior for categorization models that starts without pre-specified dimensions and learns to generalize new categories in the same way that previous categories varied. We show that without the use of pre-specified dimensions, we are able to produce generalizations that fit human data. We demonstrate that training the model on reasonable category structures produces generalization behavior that mimics that of human subjects at various ages. In addition, our trained model predicts the challenging effect of violations of the triangle inequality for similiarity judgments.

## 2   Modeling Dimensional Biases in Categorization

Models of categorization can be divided into generative and discriminative models – we will focus on generative models here and leave discriminative models for the discussion. Generative models of categorization, such as the prototype [8] and exemplar models [9, 10], assume that people learn category distributions, not just rules for discriminating between categories. In order to make a judgment of whether a new item belongs to one category or another, a comparison is made of the new item to the already existing categories, using Bayes rule with a uniform prior on the category labels,

$$P(c_n = i | x_n, \mathbf{x_{n-1}}, \mathbf{c_{n-1}}) = \frac{P(x_n | c_n = i, \mathbf{x_{n-1}}, \mathbf{c_{n-1}})}{\sum_j P(x_n | c_n = j, \mathbf{x_{n-1}}, \mathbf{c_{n-1}})} \tag{1}$$

where $x_n$ is the $n$th item and $c_n = j$ assigns that item to category $j$. The remaining items are collected in the vector $\mathbf{x_{n-1}}$ and the known labels for these items are $\mathbf{c_{n-1}}$.

For the prototype and exemplar models, the likelihood of an item belonging to a category is based on the weighted Minkowski power metric[1],

$$\sum_i \left( \sum_d w^{(d)} \left| x_n^{(d)} - R_i^{(d)} \right|^r \right)^{\frac{1}{r}} \tag{2}$$

which computes the absolute value of the power metric between the new example $x_n$ and the category representation $R_i$ for category $i$ on a dimension $d$. Integral dimensions are modeled with $r = 2$, which results in a Euclidean distance metric. The Euclidean metric has the special property that changing the basis set for the dimensions of the space does not affect the distances. Any other choice of $r$ means that the distances are affected by the basis set, and thus it must be chosen to match human judgments. Separable dimensions are modeled with either $r = 1$, the city-block metric, or $r < 1$, which no longer obeys the triangle equality [5].

Dimensional biases are also modeled in categorization by modifying the dimension weights for each dimension, $w^{(d)}$. In effect, the weights stretch or shrink the space of stimuli along each dimension so that some items are closer than others. These dimension weights are assumed to correspond to attention. To model learning of categories, it is often necessary to provide non-zero weights to only a few features early in learning and gradually shift to uniform weights late in learning [14].

These generative models of categorization have been developed to account for the different types of dimensional biases that are displayed by people, but they lack means for learning the dimensions themselves. Extensions to these classical models learn the dimension weights [15, 16], but can only learn the weights for pre-specified dimensions. If the chosen basis set did not match that used by people, then the models would be very poor descriptions of human dimensional biases. A stronger notion of between-category learning is required.

## 3   Rational Models of Categorization

Rational models of categorization view categorization behavior as the solution to a problem posed by the environment: how best to to generalize properties from one object to another. Both exemplar and prototype models can be viewed as restricted versions of rational models of categorization, which also allow interpolations between these two extreme views of representation. Anderson [11] proposed a rational model of categorization which modeled the stimuli in a task as a mixture of clusters. This model treated category labels as features, performing unsupervised learning. The model was extended to supervised learning so each category is a mixture [17],

$$P(x_\ell|\mathbf{x}_{\ell-1}, \mathbf{s}_{\ell-1}) = \sum_{k=1}^{K} P(s_\ell = k|\mathbf{s}_{\ell-1})P(x_\ell|s_\ell = k, \mathbf{x}_{\ell-1}, \mathbf{s}_{\ell-1}) \qquad (3)$$

where $x_\ell$ is the newest example in a category $i$ and $\mathbf{x}_{\ell-1}$ are the other members of category $i$. $x_\ell$ is a mixture over a set of $K$ components with the prior probability of $x_\ell$ belonging to a component depending on the component membership of the other examples $\mathbf{s}_{\ell-1}$.

Instead of a single component or a component for each previous item, the mixture model has the flexibility to choose an intermediate number of components. To make full use of this flexibility, Anderson used a nonparametric Chinese Restaurant Process (CRP) prior on the mixing weights, which allows the flexibility of having an unspecified and potentially infinite number of components (i.e., clusters) in our mixture model. The mixing proportions in a CRP are based on the number of items already included in the cluster,

$$P(s_\ell = k|\mathbf{s}_{\ell-1}) = \begin{cases} \frac{M_k}{i-1+\alpha} & \text{if } M_k > 0 \text{ (i.e., } k \text{ is old)} \\ \frac{\alpha}{i-1+\alpha} & \text{if } M_k = 0 \text{ (i.e., } k \text{ is new)} \end{cases} \qquad (4)$$

where $M_j$ is the number of objects assigned to component $k$, and $\alpha$ is the dispersion parameter. Using Equation 4, the set of assignments $s_{\ell-1}$ is built up as a simple sequential stochastic process [18] in which the order of the observations is unimportant [19].

The likelihood of belonging to a component depends on the other members of the cluster. In the case of continuous data, the components were modeled as Gaussian distributions,

$$P(x_\ell|s_\ell = k, \mathbf{x}_{\ell-1}, \mathbf{s}_{\ell-1}) = \prod_d \int_{\mu^{(d)}} \int_{\Sigma^{(d)}} N(x_\ell^{(d)}; \mu^{(d)}, \Sigma^{(d)})P(\Sigma^{(d)})P(\mu^{(d)}|\Sigma^{(d)}) \qquad (5)$$

where the mean and variance of each Gaussian distribution is given by $\mu^{(d)}$ and $\Sigma^{(d)}$ respectively. The prior for the mean was assumed to be Gaussian given the variance and the prior for the variance was a inverse-gamma distribution. The likelihood distribution for this model assumes a fixed basis set of dimensions, which must align with the separable dimensions to produce dimensional biases in generalization.

## 4    A Prior for Dimensional Learning

The rational model presented above assumes a certain basis set of dimensions, and the likelihood distributions are aligned with these dimensions. To allow the learning of the basis set, we first need multivariate versions of the prior distributions over the mean and variance parameters. For the mean parameter, we will use a multivariate Gaussian distribution and for the covariance matrix, we will use the multivariate generalization of the inverse-gamma distribution, the inverse-Wishart distribution.

The inverse-Wishart distribution has its mode at $\frac{\Sigma}{m+D+1}$, where $\Sigma$ is the mean covariance matrix parameter, $m$ is the degrees of freedom, and $D$ is the number of dimensions of the stimulus. A covariance matrix is always diagonal under some rotated version of the initial basis set. This new basis set gives the possible dimensional biases for this cluster.

However, using Gaussian distributions for each cluster, with a unimodal prior on the covariance matrix, greatly limits the patterns of generalizations that can be produced. For a diagonal covariance matrix, strong generalization along a particular dimension would be produced if the covariance matrix has a high variance along that dimension, but low variances along the remaining dimensions. Thus, this model can learn to strongly generalize along one dimension, but people often make strong generalizations along multiple dimensions [5], such as in Equation 2 when $r < 1$. A unimodal prior on covariance matrices cannot produce this behavior, so we use a mixture of inverse Wishart distributions as a prior for covariance matrices,

$$p(\Sigma_k|\mathbf{u_k}, \boldsymbol{\Phi}) = \sum_{j=1}^{J} p(u_k = j|\mathbf{u_{k-1}})p(\Sigma_k|\Phi_j, u_k = j) \tag{6}$$

where $\Sigma_k$ is the covariance parameter for the $k$th component. For simplicity, the component parameters $\Sigma_k$ are assumed i.i.d. given their class. $\Phi_j$ are the parameters of component $j$ which reflect the expected covariances generated by the $j$th inverse-Wishart distribution in the mixture. $u_k = j$ is the assignment of parameters $\Sigma_k$ to component $\Phi_j$ and the set of all other component assignments is $\mathbf{u_{k-1}}$. $\boldsymbol{\Phi}$ and $\mu_{\mathbf{k}}$ are the sets of all $\Phi_j$ and $\mu_k$. The means of categories $k$ have Gaussian priors, which depend on $\Sigma_k$, but are otherwise independent of each other.

As before, we will use a nonparametric CRP prior over the mixture weights $\mathbf{u_k}$. We now have two infinite mixtures: one that allows a category to be composed of a mixture of clusters, and one that allows the prior for the covariance matrices to be composed of a mixture of inverse-Wishart distributions. The final piece of the model is to specify $p(\Phi)$. We use another inverse-Wishart prior, but with an identity matrix for the mean parameter, so as not to bias the $\Phi_j$ components toward a particular dimension basis set. Figure 1 gives a schematic depiction of the model.

## 5    Learning the Prior

The categories we learn during development often vary along separable dimensions – and people are sensitive to this variability. The linguistic classification of nouns helps to identify categories that are fixed on one separable dimension and variable on others. Nouns can be classified into count nouns and mass nouns. Count nouns refer to objects that are discrete, such as books, shirts, and cars. Mass nouns are those that refer to objects that appear in continuous quantities, such as grass, steel, and milk. These two types of nouns show an interesting regularity: count nouns are often relatively similar in size but vary greatly in color, while mass nouns are often relatively fixed in color but vary greatly in size.

Smith [7] tested the development of children's dimensionality biases. In this study, experimenters showed participants six green circles that varied in shade and size. The discriminability judgments of adults to scale the parameters of the stimuli, so that one step in color caused the same gain in discriminability as one step in size. Participants were asked to group the stimuli into clusters according

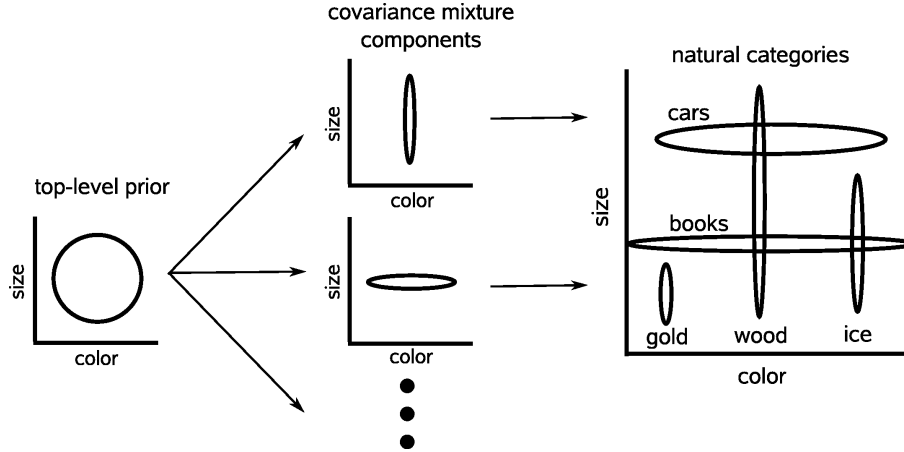

Figure 1: Schematic illustration of the hierarchical prior over covariance matrices. The top-level prior is a covariance matrix (shown as the equiprobability curves of a Gaussian) that is not biased towards any dimension. The mid level priors $\Phi_j$ are drawn from an inverse-Wishart distribution centered on the top-level prior. The $\Phi_j$ components are used as priors for the covariance matrices for clusters. The plot on the right shows some schematic examples of natural categories that tend to vary along either color or size. The covariance matrices for these clusters are drawn from an inverse-Wishart prior using one of the $\Phi_j$ components.

to their preferences, only being told that they should group the "ones that go together". The partitions of the stimuli into clusters that participants produced tended toward three informative patterns, shown in Figure 2. The Overall Similarity pattern ignores dimension and appears to result from isotropic similarity. The One-dimensional Similarity pattern is more biased towards generalizing along separable dimensions than the Overall Similarity pattern. The strongest dimensional biases are shown by the One-Dimensional Identity pattern, with the dimensional match overriding the close isotropic similarity between neighboring stimuli.

Children aged 3 years, 4 years, 5 years and adults participated in this experiment. There were ten participants in each age group, participants clustered eight problems each, and all dimension-aligned orientations of the stimuli were tested. Figure 2 shows the developmental trend of each of the informative clustering patterns. The tendency to cluster according to Overal Similarity decreased with age, reflecting a reduced influence of isotropic similarity. Clustering according One-dimensional Similarity increased from 3-year-olds to 5-year-olds, but adults produced few of these patterns. The percentage of One-dimensional Identity clusterings increased with age, and was the dominant response for adults, supporting the idea that strong dimensional biases are learned.

We trained our model with clusters that were aligned with the dimensions of size and color. Half of the clusters varied strongly in shape and weakly in size, while the other half varied strongly in size and weakly in shape. The larger standard deviation of the distribution that generated the training stimuli was somewhat smaller than the largest distance between stimuli in the Smith experiment, while the smaller standard deviation in the distribution that generated the training stimuli was much smaller than the smallest distance between Smith stimuli. The two dispersion parameters were set to 1, the degrees of freedom for all inverse-Wishart distributions were set to the number of dimensions plus 1, and 0.01 was used for the scale factor for the mean parameters of the inverse-Wishart distributions[2].

Inference in the model was done as a combination of the Gibbs sampling and Metropolis-Hastings algorithms. The assignments of data points to clusters in each class were Gibbs sampled conditioned on the cluster assignments to inverse-Wishart components and the parameters of those components, $\Phi_j$. Following a complete pass of the assignments of data points to clusters, we then Gibbs sampled the assignments of the cluster covariance parameters $\Sigma_k$ to components of the inverse-Wishart

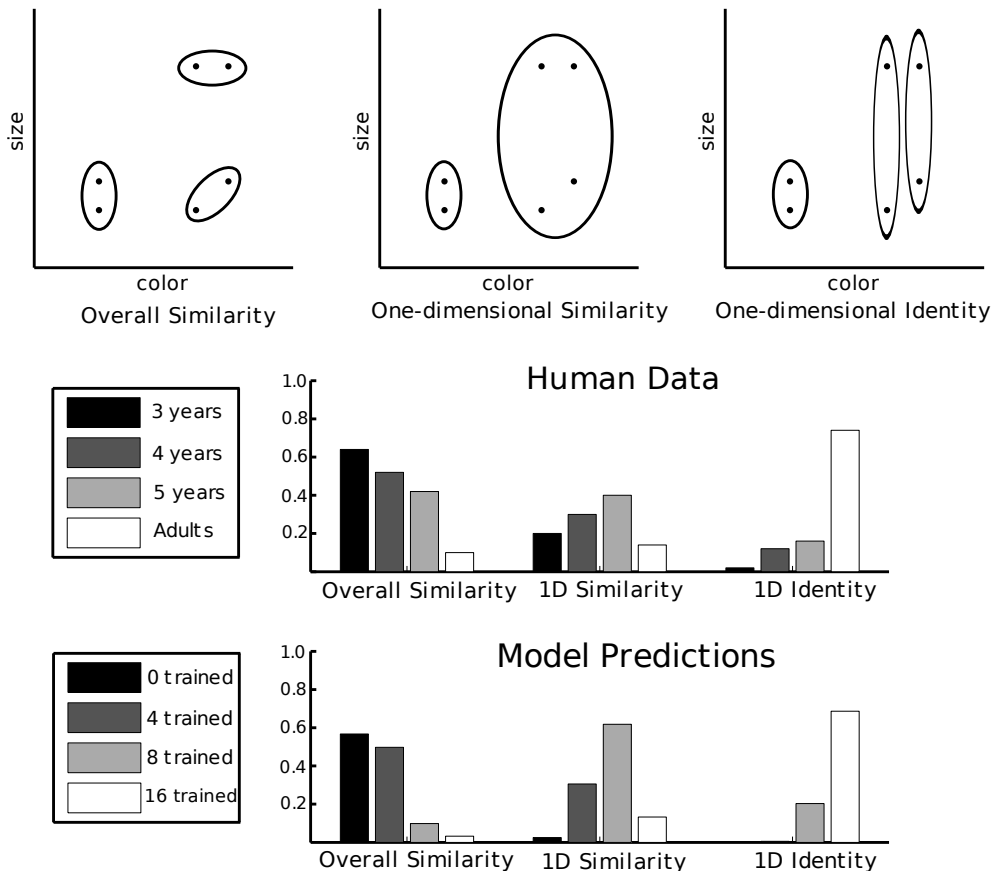

Figure 2: Experiment 2 of Smith [7]. In a free categorization task, the stimuli marked by dots in the top row were grouped by participants. The three critical partitions are shown as circles in the top row of plots. The top bar graph displays the developmental trends for each of the critical partitions. The bottom bar graph displays the trend as the model is trained on a larger number of axis-aligned clusters.

mixture prior. After a pass of this mid-level sampling, we resampled $\Phi_j$, the parameters of the inverse-Wishart components, and the prior expected means of each cluster. This sampling was done using Metropolis-Hastings, with the non-symmetric proposals made from a separate inverse-Wishart distribution. A large finite Dirichlet distribution was used to approximate $p(U)$. Given the learned $\Phi$ and $\mathbf{u_k}$, the predicted probabilities for the Smith experiment were computed exactly.

The predictions of our model as a result of training are shown in Figure 2. The model was trained on 0, 2, 4, 8, and 16 axis-aligned clusters in an unsupervised fashion. For the all three patterns, the model shows the same developmental trajectory as human data. Overall Similarity decreases with the number of trained categories, One-dimensional Similarity increases and then decreases, and One-dimensional Identity patterns are overwhelmingly produced by the fully trained model. The probabilities plotted in the figure are the predicted posterior of only the partitions that exactly matched the informative patterns, out of all 203 possible partitions, showing that the patterns in Figure 2 dominated the model's predictions as they dominated the participants' responses in the free categorization task.

## 6 Generalization Gradients

Standard models of categorization, such as the prototype or exemplar model, have a variety of mechanisms for producing the dimensional biases seen in experiments with adults. We propose a very different explanation for these dimensional biases. In this section we plot generalization gradients,

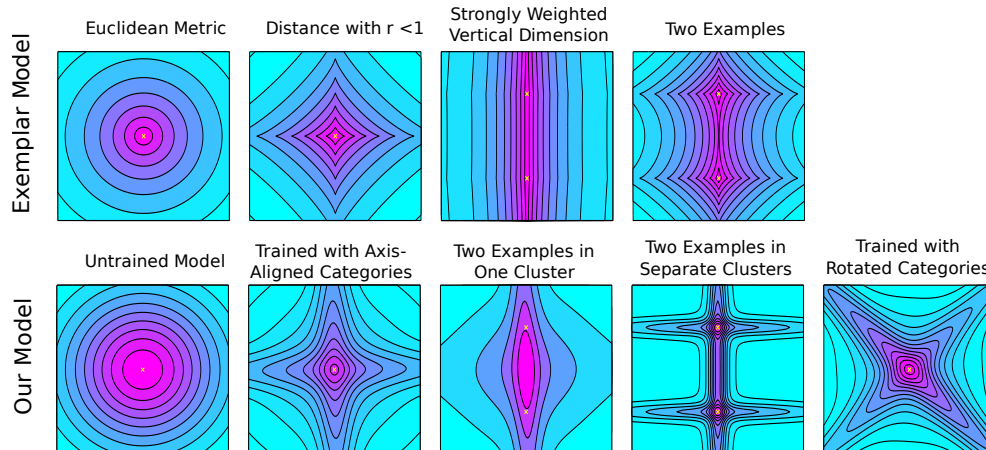

Figure 3: Generalization gradients of the exemplar model and the posterior predictive distribution of the model presented in this paper. The dots are the stimuli.

which provide a good feel for how the priors we propose match with the mechanisms used in earlier models across a variety of conditions.

Generalizations of single items are studied by collecting similarity ratings. In this task, participants judge the similarity of two items. In standard models of categorization, similarity ratings are modeled mainly by the exponent in the Minkowski power metric (Equation 2). For rational models, similarity ratings can be modeled as the posterior predictive probability of one item, given the second item [20]. The first two columns of Figure 3 give a comparison between the exemplar model and the model we propose for similarity ratings. The central dot is a particular stimulus and the color gradient shows the predicted similarity ratings of all other stimuli. For integral dimensions, a Euclidean metric ($r = 2$) is used in the exemplar model, which the model we propose matches if it has not been trained on dimension-aligned categories.

For separable categories, the exemplar model usually uses a city-block metric ($r = 1$) [10]. However, experimental evidence shows that dimensions have an even stronger effect than predicted by a city-block metric. In experiments to test violations of the triangle equality, Tversky and Gati [5] showed that the best fitting exponent for similarity data is often $r < 1$. The model we propose can produce this type of similarity prediction by using a prior that is a mixture of covariance matrices, in which each component of the mixture generalizes strongly along one dimension. In a category of one item, which is the case when making similarity judgments with the posterior predictive distribution, it is uncertain which covariance component best describes the category. This uncertainty results in a generalization gradient that imitates an exponent of $r < 1$ using Gaussian distributions. As a result, our proposed model predicts violations of the triangle inequality if it has been trained on a set of clusters in which some vary strongly along one dimension and others vary strongly along another dimension. A comparison between this generalization gradient and the exemplar model is shown in the second column of Figure 3.

The second mechanism for dimensional biases in standard models of categorization is selective attention. Selective attention is used to describe biases that occur in categorization experiments, when many items are trained in each category. These biases are implemented in the exemplar model as weights along each dimension, and early in learning there are usually large weights on a small number of separable dimensions [14, 21]. Our proposed model does not have a mechanism for selective attention, but provides a rational explanation for this effect in terms of the strong sampling assumption [13]. If two items are assumed to come from the same cluster, then generalization tends to be along a single dimension that has varied during training (third column of Figure 3). However, if two items are inferred to belong to different clusters, then the generalization gradient corresponds to additive similarity without selective attention (fourth column of Figure 3).

We have shown that the model we have proposed can reproduce the key generalization gradients of the exemplar and prototype models. The important difference between our model of dimensional

biases and these standard categorization models is that we learn basis set for dimensional biases, assuming these dimensions have proven to be useful for predicting category structure in the past. Other models must have these dimensions pre-specified. To show that our model is not biased towards a particular basis set, we rotated the training stimuli 45 degrees in space. The resulting posterior predictive distributions in Figure 3 extendend in the same direction as the rotated training categories varied.

## 7    Discussion

The approach to dimensional biases we have outlined in this paper provides a single explanation for dimensional biases, in contrast to standard models of categorization, such as exemplar and prototype models. These standard models of categorization assume two distinct mechanisms for producing dimensional biases: a Minkowski metric exponent, and attentional weights for each dimension. In our approach, biases in both similarity judgments and categorization experiments are produced by learning covariance matrices that are shared between clusters. For similarity judgments, the single item does not give information about which covariance mixture component was used to generate it. This uncertainty produces similarity judgments that would be best fit with an Minkowski exponent of $r < 1$. For category judgments, the alignment of the items along a dimension allows the generating covariance mixture component to be inferred, so the judgments will show a bias like that of attentional weights to the dimensions. The difference between tasks drives the different types of dimensional biases in our approach.

We propose that people learn more complex cross-category information than most previous approaches do. Attention to dimensions is learned in connectionist models of categorization by finding the best single set of weights for each dimension in a basis set [15, 16], or by cross-category learning in a Bayesian approach [22]. A more flexible approach is used in associative models of categorization, which allow for different patterns of generalizations for different items. One associative model used a Hopfield network to predict different generalizations for solid and non-solid objects [23]. A hierarchical Bayesian model with very similar properties to this associative model motivated this result from cross-category learning [24]. The key difference between all these models and our proposal is that they use only a single strong dimensional bias for each item, while we use multiple latent strong dimensional biases for each item, which is needed for modeling both similarity and categorization dimensional biases with a single explanation. The only previous approach we are aware of that learns such complex cross-category information is a Bayesian rule-based model of categorization [25].

The main advantage of our approach over many other models of categorization is that we learn the basis set of dimensions that can display dimensional biases. Our model learns the basis the same way people do, from categories in the environment (as opposed to fitting to human similarity or category judgements). We begin with a feature space of stimuli in which physically similar items are near to each other. Using a version of the Transformed Dirichlet Process [26], a close relation to the Hierarchical Dirichlet Process previously proposed as a unifying model of categorization [17], a mixture of covariance matrices are learned from environmentally plausible training data. Most other models of categorization, including exemplar models [10], prototype models [8], rule-based discriminative models [27], as well as hierarchical Bayesian models for learning features [24, 22] and Bayesian rule-based models [25] all must have a pre-specified basis set.

## 8    Summary and Conclusions

People generalize categories in two ways: they generalize to stimuli with parameters near to the category and generalize to stimuli that match along separable dimensions. Existing models of categorization must assume the dimensions to produce human-like generalization performance. Our model learns these dimensions from the data: starting with an unbiased prior, the dimensions that categories vary along are learned to be dimensions important for generalization. After training the model with categories intended to mirror those learned during development, our model reproduces the trajectory of generalization biases as children grow into adults. Using this type of approach, we hope to better tie models of human generalization to the natural world to which we belong.

## Footnotes

[1]For an exemplar model, $R_i^{(d)}$ is each example in $\mathbf{x_{n-1}}$, while for the prototype model, it is the single average of $\mathbf{x_{n-1}}$.

[2]The general pattern of the results was only weakly dependent on the parameter settings, but unsupervised learning of the clusters required a small value of the scale factor.

# References

[1] E. Rosch, C. B. Mervis, W. D. Gray, D. M. Johnson, and P. Boyes-Braem. Basic objects in natural categories. *Cognitive Psychology*, 8:382–439, 1976.

[2] R. N. Shepard. Toward a universal law of generalization for psychological science. *Science*, 237:1317–1323, 1987.

[3] W. R. Garner. *The processing of information and structure*. Erlbaum, Hillsdale, NJ, 1974.

[4] J. K. Kruschke. Human category learning: implications for backpropagation models. *Connection Science*, 5:3–36, 1993.

[5] A. Tversky and I. Gati. Similarity, separability and the triangular inequality. *Psychological Review*, 93:3–22, 1982.

[6] L. B. Smith and Kemler D. G. Developmental trends in free classification: Evidence for a new conceptualization of perceptual development. *Journal of Experimental Child Psychology*, 24:279–298, 1977.

[7] L. B. Smith. A model of perceptual classification in children and adults. *Psychological Review*, 96:125–144, 1989.

[8] S. K. Reed. Pattern recognition and categorization. *Cognitive Psychology*, 3:393–407, 1972.

[9] D. L. Medin and M. M. Schaffer. Context theory of classification learning. *Psychological Review*, 85:207–238, 1978.

[10] R. M. Nosofsky. Attention, similarity, and the identification-categorization relationship. *Journal of Experimental Psychology: General*, 115:39–57, 1986.

[11] J. R. Anderson. The adaptive nature of human categorization. *Psychological Review*, 98(3):409–429, 1991.

[12] D. J. Navarro. From natural kinds to complex categories. In *Proceedings of CogSci*, pages 621–626, Mahwah, NJ, 2006. Lawrence Erlbaum.

[13] J. B. Tenenbaum and T. L. Griffiths. Generalization, similarity, and Bayesian inference. *Behavioral and Brain Sciences*, 24:629–641, 2001.

[14] M. K. Johansen and T. J. Palmeri. Are there representational shifts in category learning? *Cognitive Psychology*, 45:482–553, 2002.

[15] John K. Kruschke. Alcove: An exemplar-based connectionist model of category learning. *Psychological Review*, 99:22–44, 1992.

[16] B. C. Love, D. L. Medin, and T. M. Gureckis. SUSTAIN: A network model of category learning. *Psychological Review*, 111:309–332, 2004.

[17] T. L. Griffiths, K. R. Canini, A. N. Sanborn, and D. J. Navarro. Unifying rational models of categorization via the hierarchical dirichlet process. In R. Sun and N. Miyake, editors, *Proceedings CogSci*, 2007.

[18] D. Blackwell and J. MacQueen. Ferguson distributions via Polya urn schemes. *The Annals of Statistics*, 1:353–355, 1973.

[19] D. Aldous. Exchangeability and related topics. In *École d'été de probabilités de Saint-Flour, XIII—1983*, pages 1–198. Springer, Berlin, 1985.

[20] T. L. Griffiths, M. Steyvers, and J. B. Tenenbaum. Topics in semantic representation. *Psychological Review*, 114:211–244, 2007.

[21] R. M. Nosofsky and S. R. Zaki. Exemplar and prototype models revisted: response strategies, selective attention, and stimulus generalization. *Journal of Experimental Psychology: Learning, Memory, and Cognition*, 28:924–940, 2002.

[22] A. Perfors and J.B. Tenenbaum. Learning to learn categories. In *Proceedings of CogSci*, 2009.

[23] E. Colunga and L. B. Smith. From the lexicon to expectations about kinds: a role for associative learning. *Psychological Review*, 112, 2005.

[24] C. Kemp, A. Perfors, and J. B. Tenenbaum. Learning overhypotheses with hierarchical Bayesian models. *Developmental Science*, 10:307–321, 2007.

[25] N. D. Goodman, J. B. Tenenbaum, J. Feldman, and T. L. Griffiths. A rational analysis of rule-based concept learning. *Cognitive Science*, 32:108–154, 2008.

[26] E. Sudderth, A. Torralba, W. Freeman, and A. Willsky. Describing visual scenes using transformed dirichlet processes. In *Neural Information Processing Systems NIPS*, 2005.

[27] R. M. Nosofsky and T. J. Palmeri. A rule-plus-exception model for classifying objects in continuous-dimension spaces. *Psychonomic Bulletin & Review*, 5:345–369, 1998.

